# Performance of Connectionist Learning Algorithms on 2-D SIMD Processor Arrays

**Fernando J. Núñez\*  and  Jose A.B. Fortes**
School of Electrical Engineering
Purdue University
West Lafayette, IN 47907

## ABSTRACT

The mapping of the back-propagation and mean field theory learning algorithms onto a generic 2-D SIMD computer is described. This architecture proves to be very adequate for these applications since efficiencies close to the optimum can be attained. Expressions to find the learning rates are given and then particularized to the DAP array procesor.

## 1  INTRODUCTION

The digital simulation of connectionist learning algorithms is flexible and accurate. However, with the exception of very small networks, conventional computer architectures spend a lot of time in the execution of simulation software. Parallel computers can be used to reduce the execution time. Vector-pipelined, multiprocessors, and array processors are some of the most important classes of parallel computers[3]. Connectionist or neural net (NN) learning algorithms have been mapped onto all of them.

The focus of this contribution is on the mapping of the back-propagation (BP) and mean field theory (MFT) learning algorithms onto the subclass of SIMD computers with the processors arranged in a square two-dimensional mesh and interconnected by nearest-neighbor links.

The material is organized as follows. In section 2, the execution cost of BP and MFT on sequential computers is found. Two-dimensional SIMD processor arrays are described in section 3, and the costs of the two dominanting operations in the simulations are derived. In section 4 the mapping of BP and MFT is commented

\* Current address: Motorola Inc., 1301 E Algonquin Rd., Schaumburg, IL 60196

and expressions for the learning rates are obtained. These expressions are particularized to the DAP computer in section 5. Section 6 concludes this work.

## 2 BACK-PROPAGATION AND MEAN FIELD THEORY

In this paper, two learning algorithms: BP[7] and MFT[4]; and 3-layer nets are considered. The number of neurons in the input, hidden, and output layer is I, H, and O respectively. BP has been used in many applications. Probably, NETtalk[8] is the best known. MFT can also be used to learn arbitrary mappings between two sets, and remarkably, to find approximate solutions to hard optimization problems much more efficiently than a Boltzmann Machine does[4,5].

The output of a neuron $i$ will be denoted as $v_i$ and called *value:* $v_i = f\left(\sum_{j \neq i} a_{ij} v_j - \theta_i\right)$. The summation represents the net input received and will be called *activation*. The neuron thresold is $\theta_i$. A sigmoid-like function $f$ is applied to find the value. The weight of the link from neuron j to neuron i is $a_{ij}$. Since input patterns are the values of the I layer, only neuron values and activations of the H and O layers must be computed. In BP, the activation error and the value error of the H and O layers are calculated and used to change the weights.

In a conventional computer, the execution time of BP is approximately the time spent in finding the activations, back-propagating the activation error of the O layer, and modifying the I-H and H-O weights. The result is: $(2I + 3O)Ht_m$, where $t_m$ is the time required to perform a multiply/accumulate operation. Since the net has $(I + O)H$ connections, the learning rate in connections per second is:

$$\mathcal{L}_{BP} = \frac{I + O}{(2I + 3O)t_m} \quad CPS$$

In the MFT algorithm, only from the neuron values in equilibrium at the end of the clamped and free annealing phases we can compute the weight increments. It is assumed that in both phases there are $A$ annealing temperatures and that $E$ iterations are enough to reach equilibrium at each temperature[4,5]. With these changes, MFT is now a deterministic algorithm where the annealing phases are composed of $AE$ sweeps. The MFT execution time can be approximated by the time spent in computing activations in the annealing loops. Taking into account that in the clamped phase only the H layer is updated, and that in the free phase both, the H and O layers change their values, the MFT learning performance is found to be:

$$\mathcal{L}_{MFT} = \frac{\mathcal{L}_{BP}}{AE} \quad CPS$$

MFT is $AE$ times more expensive than BP. However, the learning qualities of both algorithms are different and such a direct comparison is simplistic.

## 3  2-D SIMD PROCESSOR ARRAYS

Two-dimensional single instruction multiple data stream (2-D SIMD) computers are very efficient in the simulation of NN learning algorithms. They can provide massive parallelism at low cost. An SIMD computer is an array of processing elements (PEs) that execute the same instruction in each cycle. There is a single control unit that broadcasts instructions to all the PEs. SIMD architectures operate in a synchronous, lock-step fashion[3]. They are also called *array procesors* because their *raison d'être* is to operate on vectors and matrices.

Example SIMD computers are the Illiac-IV, the Massively Parallel Processor (MPP), the Connection Machine (CM), and the Distributed Array Processor (DAP). With the exception of the CM, whose PE interconnection topology is a hypercube, the other three machines are 2-D SIMD arrays because their PEs are interconnected by a 2-D mesh with wrap-around links (figure 1).

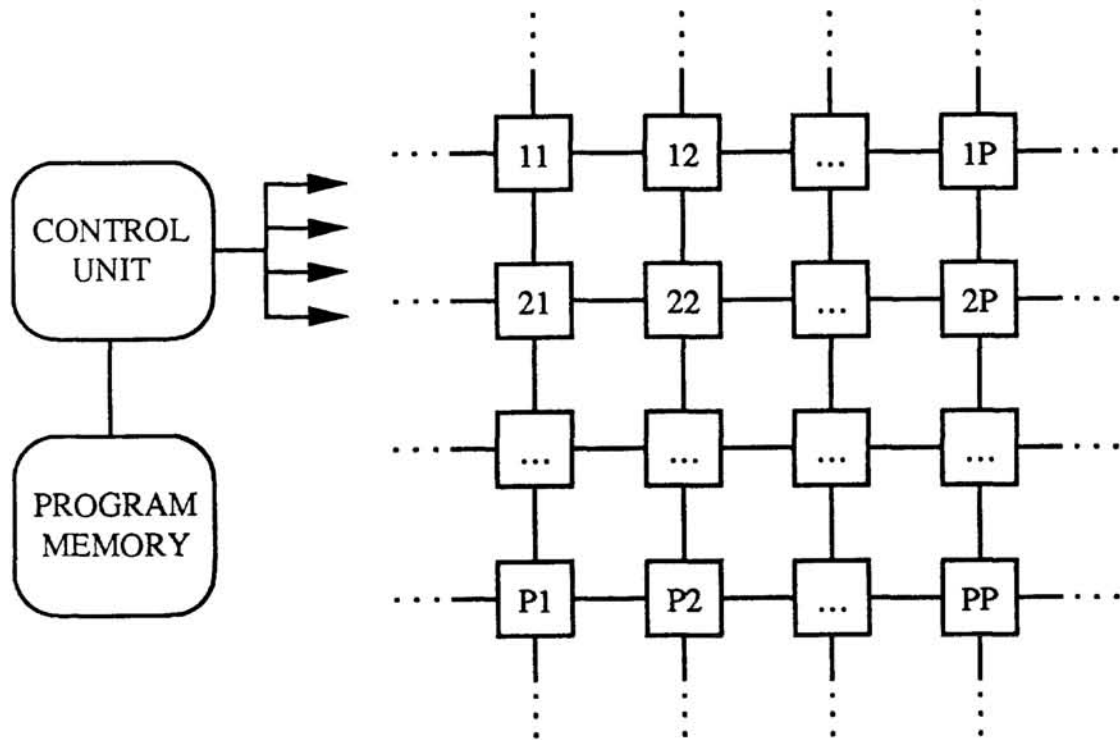

**Figure 1:** A 2-D SIMD Processor Array

Each PE has its own local memory. The instruction has an address field to access it. The array memory space can be seen as a 3-D volume. This volume is generated by the PE plane, and the depth is the number of memory words that each PE can address. When the control unit issues an address, a plane of the memory volume is being referenced. Then, square blocks of $P{\times}P$ elements are the natural addressing unit of 2-D SIMD processor arrays. There is an activity bit register in each PE to disable the execution of instructions. This is useful to perform operations with a subset of the PEs. It is assumed that there is no

overlapping between data processing an data moving operations. In other words, PEs can be either performing some operation on data (this includes accessing the local memory) or exchanging data with other processors.

## 3.1   MAPPING THE TWO BASIC OPERATIONS

It is characteristic of array processors that the way data is allocated into the PEs memories has a very important effect on performance. For our purposes, two data structures must be considered: vectors and matrices. The storage of vectors is illustrated in figure 2-a. There are two modes: row and column. A vector is split into $P$-element subvectors stored in the same memory plane. Very large vectors will require two or more planes. The storage of matrices is also very simple. They must be divided into square $P \times P$ blocks (figure 2-b). The shading in figure 2 indicates that, in general, the sizes of vectors and matrices do not fit the array dimensions perfectly.

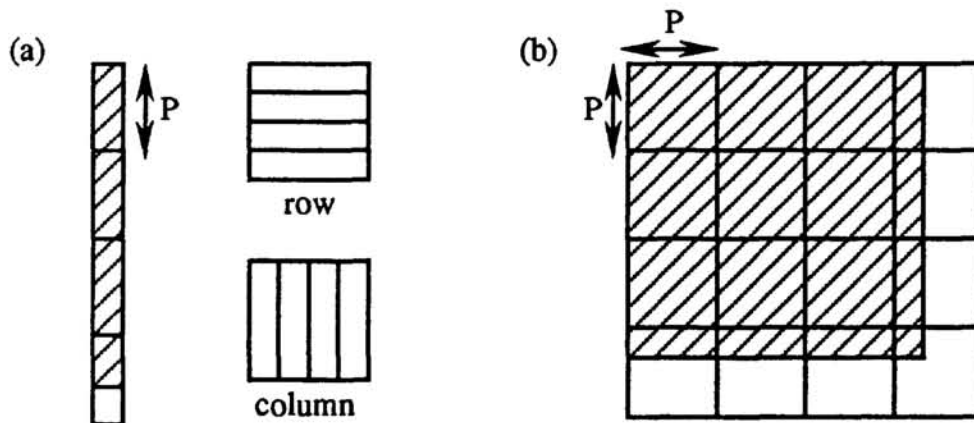

Figure 2: (a) Vector and (b) Matrix Storage

The execution time of BP and MFT in a 2-D SIMD computer is spent, almost completely, in matrix-vector multiply (MVM) and vector outer multiply/accumulate (VOM) operations. They can be decomposed in the following simpler operations involving $P \times P$ blocks.

a) Addition (+): $C = A + B$ such that $c_{ij} = a_{ij} + b_{ij}$.
b) Point multiply/accumulate (·):  $C = C + A \cdot B$ such that $c'_{ij} = c_{ij} + a_{ij}b_{ij}$.
c) Unit rotation:  The result block has the same elements than the original, but rotated one place in one of the four possible directions (N, E, W, and S).
d) Row (column) broadcast:  The result of the row (column) broadcast of a vector $z$ stored in row (column) mode is a block $X$ such that $x_{ij} = x_j \ (= x_i)$.

The time required to execute a, b, c, and d will be denoted as $t_a$, $t_m$, $t_r$, and $t_b$ respectively. Next, let us see how the operation $y = Ax$ (MVM) is decomposed in simpler steps using the operations above. Assume that $x$ and $y$ are $P$-element vectors, and $A$ is a $P \times P$ block.

1) Row-broadcast vector $x$.
2) Point multiply $Y = A \cdot X$.
3) Row addition of block Y, $y_i = \sum\limits_{j=1}^{P} y_{ij} = \sum\limits_{j=1}^{P} a_{ij}x_j$. This requires $\left\lceil \log_2 P \right\rceil$ steps. In each step multiple rotations and one addition are performed. Figure 3 shows how eight values in the same row are added using the recursive doubling technique. Note that the number of rotations doubles in each step. The cost is: $Pt_r + \log_2 Pt_a$. Row addition is an inefficient operation because of the large cost due to communication. Fortunately, for larger data its importance can be diminished by using the scheduling described nextly.

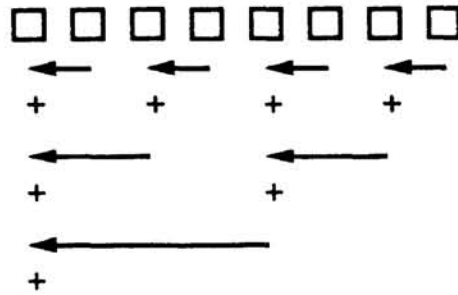

**Figure 3:** Recursive Doubling

Suppose that $x$, $y$, and $A$ have dimensions $m = MP$, $n = NP$, and $n \times m$ respectively. Then, $y = Ax$ must be partitioned into a sequence of non-partitioned block operations as the one explained above. We can write:

$$y^i = \sum_{j=1}^{M} A^{ij} x^j = \sum_{j=1}^{M} (A^{ij} \cdot X^j)u = \left( \sum_{j=1}^{M} A^{ij} \cdot X^j \right)u$$

In this expression, $y^i$ and $x^j$ represent the $i$-th and $j$-th $P$-element subvector of $y$ and $x$ respectively, and $A^{ij}$ is the $P \times P$ block of $A$ with indices $i$ and $j$. Block $X^j$ is the result of row-broadcasting $x^j$ ($x$ is stored in row mode.) Finally, $u$ is a vector with all its $P$-elements equal to 1. Note that in the second term $M$ column additions are implicit, while only one is required in the third term because blocks instead of vectors are accumulated. Since $y$ has $N$ subvectors, and the $M$ subvectors of $x$ are broadcast only once, the total cost of the MVM operation is:

$$NMt_m + N(Pt_r + \log_2 Pt_a) + Mt_b$$

After a similar development, the cost of the VOM ( $A' = A + yx^T$ ) operation is:

$$NMt_m + (N + M)t_b$$

If the number of neurons in each layer is not an integer multiple of $P$, the storage and execution efficiencies decrease. This effect is less important in large networks.

# 4   LEARNING RATES ON 2-D SIMD COMPUTERS

## 4.1   BACK-PROPAGATION

The neuron values, activations, value errors, activation errors, and thresolds of the H and O layers are organized as vectors. The weights are grouped into two matrices: I-H and H-O. Then, the scalar operations of the original algorithm are transformed into matrix-vector operations.

From now on, the size of the input, hidden, and output layers will be $IP$, $HP$, and $OP$. As commented before, the execution time is mostly spent in computing activations, values, their errors, and in changing the weights. To compute activations, and to back-propagate the activation error of the O layer MVM operations are performed. The change of weights requires VOM operations. After substituting the expressions of the previous section, the time required to learn a pattern simulating BP on a 2-D SIMD computer is:

$$(2I + 3O)Ht_m + (2I + 3H + 2O)t_b + (2H + O)(Pt_r + \log_2 Pt_a)$$

The time spent in data communication is given by the factors in $t_r$ and $t_b$. The larger they are, the smaller is the efficiency. For array processors with fast broadcast facilities, and for nets large enough in terms of the array dimensions, the efficiency grows since a smaller fraction of the total execution time is dedicated to moving data. Since the net has $(I + O)HP^2$ connections, the learning rate is $P^2$ times greater than using a single PE:

$$\mathcal{L}_{SIMD-BP} = \frac{(I + O)P^2}{(2I + 3O)t_m} \quad CPS$$

## 4.2   MEAN FIELD THEORY

The operations outside the annealing loops can be neglected with small error. In consequence, only the computation of activations in the clamped and free annealing phases is accounted for:

$$AE((2I + 3O)Ht_m + (2I + H + 2O)t_b + (2H + O)(Pt_r + \log_2 Pt_a))$$

Under the same favorable conditions above mentioned, the learning rate is:

$$\mathcal{L}_{SIMD-MFT} = \frac{(I + O)P^2}{AE(2I + 3O)t_m} \quad CPS$$

## 5   LEARNING PERFORMANCE ON THE DAP

The DAP is a commercial 2-D SIMD processor array developed by ICL. It is a massively parallel computer with bit-level PEs built around a single-bit full adder. In addition to the 2-D PE interconnection mesh, there are row and column broadcast buses that allow the direct transfer of data from any processor row or column to an edge register. Many instructions require a single clock cycle leading to very efficient codings of loop bodies. The DAP-510 computer features $2^5 \times 2^5$ PEs with a maximum local memory of 1Mbit per PE. The DAP-610 has $2^6 \times 2^6$ PEs, and the maximum local memory is 64Kbit. The clock cycle in both machines is 100 ns[1].

With bit-level processors it is possible to tailor the precision of fixed-point computations to the minimum required by the application. The costs in cycles required by several basic operations are given below. These expressions are function of the number of bits of the operands, that has been assumed to be the same for all of them: $b$ bits.

The time required by the DAP to perform a block addition, point multiplication/accumulation, and broadcast is $t_a = 2b$, $t_m = 2b^2$, and $t_b = 8b$ clock cycles respectively. On the other hand, $P + 2b \log_2 P$ cycles is the duration of a row addition. Let us take $b = 8$ bits, and $AE = 24$. This values have been found adequate in many applications. Then, the maximum learning rates of the DAP-610 ($P = 64$) are:

$$\text{BP: } 100\text{-}160 \; MCPS \qquad \text{MFT: } 4.5\text{-}6.6 \; MCPS$$

where $MCPS \equiv 10^6 CPS$. These figures are 4 times smaller for the DAP-510. It is worth to mention that the performance decreases quadratically with $b$. The two learning rates of each algorithm correspond to the worst and best case topology.

### 5.1   EXAMPLES

Let us consider a one-thousand neuron net with 640, 128, and 256 neurons in the input, hidden, and output layer. For the DAP-610 we have $I = 10$, $H = 2$, and $O = 4$. The other parameters are the same than used above. After substituting, we see that the communication costs are less than 10% of the total, demonstrating the efficiency of the DAP in this type of applications. The learning rates are:

$$\text{BP: } 140 \; MCPS \qquad \text{MFT: } 5.8 \; MCPS$$

NETtalk[10] is frequently used as a benchmark in order to compare the performance achieved on different computers. Here, a network with similar dimensions is considered: 224 input, 64 hidden, and 32 output neurons. These dimensions fit perfectly into the DAP-510 since $P = 32$. As before, a data precision of 8 bits has been taken. However, the fact than the input patterns are binary has been exploited to obtain some savings.

The performance reached in this case is 50 *MCPS*. Even though NETtalk is a relatively small network, only 30% of the total execution time is spent in data communication. If the DAP-610 were used, somewhat less than 200 *MCPS* would be learnt since the output layer is smaller than $P$ what causes some inefficiency.

Finally, BP learning rates of the DAP-610 with 8- and 16-bit operands are compared to those obtained by other machines below[2,6]:

| COMPUTER | MCPS |
| --- | --- |
| VAX 780 | 0.027 |
| CRAY-2 | 7 |
| CM (65K PEs) | 13 |
| DAP-610 (8 bits) | 100-160 |
| DAP-610 (16 bits) | 25-40 |

# 6  CONCLUSIONS

Two-dimensional SIMD array processors are very adequate for the simulation of connectionist learning algorithms like BP and MFT. These architectures can execute them at nearly optimum speed if the network is large enough, and there is full connectivity between layers. Other much more costly parallel architectures are outperformed.

The mapping approach described in this paper can be easily extended to any network topology with dense blocks in its global interconnection matrix. However, it is obvious that 2-D SIMD arrays are not a good option to simulate networks with random sparse connectivity.

**Acknowledgements**

This work has been supported by the Ministry of Education and Science of Spain.

**References**

[1] (1988) AMT DAP Series, Technical Overview. *Active Memory Technology.*

[2] G. Blelloch & C. Rosenberg. (1987) Network Learning on the Connection Machine. *Proc. 10th Joint Conf. on Artificial Intelligence,* IJCA Inc.

[3] K. Hwang & F. Briggs. (1984) *Computer Architecture and Parallel Processing,* McGraw-Hill.

[4] C. Peterson & J. Anderson. (1987) A Mean Field Theory Learning Algorithm for Neural Networks. *Complex Systems,* 1:995-1019.

[5] C. Peterson & B. Soderberg. (1989) A New Method For Mapping Optimization Problems onto Neural Networks. *Int'l J. of Neural Systems,* 1(1):3-22.

[6] D. Pomerleau, G. Gusciora, D. Touretzky & H.T. Kung. (1988) Neural Network Simulation at Warp Speed: How We Got 17 Million Connections per Second. *Proc. IEEE Int'l Conf. on Neural Networks,* II:143-150.

[7] D. Rumelhart, G. Hinton & R. Williams. (1986) Learning Representations by Back-Propagating Errors. *Nature,* (323):533-536.

[8] T. Sejnowski & C. Rosenberg. (1987) Parallel Networks that Learn to Pronounce English Text. *Complex Systems,* 1:145-168.